# Strategy Grafting in Extensive Games

**Kevin Waugh**
waugh@cs.cmu.edu
Department of Computer Science
Carnegie Mellon University

**Nolan Bard, Michael Bowling**
{nolan,bowling}@cs.ualberta.ca
Department of Computing Science
University of Alberta

## Abstract

Extensive games are often used to model the interactions of multiple agents within an environment. Much recent work has focused on increasing the size of an extensive game that can be feasibly solved. Despite these improvements, many interesting games are still too large for such techniques. A common approach for computing strategies in these large games is to first employ an abstraction technique to reduce the original game to an abstract game that is of a manageable size. This abstract game is then solved and the resulting strategy is played in the original game. Most top programs in recent AAAI Computer Poker Competitions use this approach. The trend in this competition has been that strategies found in larger abstract games tend to beat strategies found in smaller abstract games. These larger abstract games have more expressive strategy spaces and therefore contain better strategies. In this paper we present a new method for computing strategies in large games. This method allows us to compute more expressive strategies without increasing the size of abstract games that we are required to solve. We demonstrate the power of the approach experimentally in both small and large games, while also providing a theoretical justification for the resulting improvement.

## 1 Introduction

Extensive games provide a general model for describing the interactions of multiple agents within an environment. They subsume other sequential decision making models such as finite horizon MDPs, finite horizon POMDPs, and multiagent scenarios such as stochastic games. This makes extensive games a powerful tool for representing a variety of complex situations. Moreover, it means that techniques for computing strategies in extensive games are a valuable commodity that can be applied in many different domains. The usefulness of the extensive game model is dependent on the availability of solution techniques that scale well with respect to the size of the model. Recent research, particularly motivated by the domain of poker, has made significant developments in scalable solution techniques. The classic linear programming techniques [5] can solve games with approximately $10^7$ states [1], while more recent techniques [2, 9] can solve games with over $10^{12}$ states.

Despite the improvements in solution techniques for extensive games, even the motivating domain of two-player limit Texas Hold'em is far too large to solve, as the game has approximately $10^{18}$ states. The typical solution to this challenge is abstraction [1]. Abstraction involves constructing a new game that is tractably sized for current solution techniques, but restricts the information or actions available to the players. The hope is that the abstract game preserves the important strategic structure of the game, and so playing a near equilibrium solution of the abstract game will still perform well in the original game. In poker, employed abstractions include limiting the possible betting sequences, replacing all betting in the first round with a fixed policy [1], and, most commonly, by grouping the cards dealt to each player into *buckets* based on a strength metric [4, 9].

With these improvements in solution techniques, larger abstract games have become tractable, and therefore increasingly fine abstractions have been employed. Because a finer abstraction can rep-

resent players' information more accurately and provide a more expressive space of strategies, it is generally assumed that a solution to a finer abstraction will produce stronger strategies for the original game than those computed using a coarser abstraction. Although this assumption is in general not true [7], results from the AAAI Computer Poker Competition [10] have shown that it does often hold: near equilibrium strategies with the largest expressive power tend to win the competition.

In this paper, we increase the expressive power of computable strategies without increasing the size of game that can be feasibly solved. We do this by partitioning the game into tractably sized sub-games called grafts, solving each independently, and then combining the solutions into a single strategy. Unlike previous, subsequently abandoned, attempts to solve independent sub-games [1, 3], the grafting approach uses a base strategy to ensure that the grafts will mesh well as a unit. In fact, we prove that grafted strategies improve on near equilibrium base strategies. We also empirically demonstrate this improvement both in a small poker game as well as limit Texas Hold'em.

## 2   Background

Informally, an extensive game is a game tree where a player cannot distinguish between two histories that share the same information set. This means a past action, from either chance or another player, is not completely observed, allowing one to model situations of imperfect information.

**Definition 1 (Extensive Game)** *[6, p. 200] A finite extensive game with imperfect information is denoted $\Gamma$ and has the following components:*

- *A finite set $N$ of **players**.*
- *A finite set $H$ of sequences, the possible **histories** of actions, such that the empty sequence is in $H$ and every prefix of a sequence in $H$ is also in $H$. $Z \subseteq H$ are the **terminal histories**. No sequence in $Z$ is a strict prefix of any sequence in $H$. $A(h) = \{a : (h, a) \in H\}$ are the actions available after a non-terminal history $h \in H \setminus Z$.*
- *A **player function** $P$ that assigns to each non-terminal history a member of $N \cup \{c\}$, where $c$ represents chance. $P(h)$ is the player who takes an action after the history $h$. Let $H_i$ be the set of histories where player $i$ chooses the next action.*
- *A function $f_c$ that associates with every history $h \in H_c$ a probability distribution $f_c(\cdot|h)$ on $A(h)$. $f_c(a|h)$ is the probability that $a$ occurs given $h$.*
- *For each player $i \in N$, a **utility function** $u_i$ that assigns each terminal history a real value. $u_i(z)$ is rewarded to player $i$ for reaching terminal history $z$. If $N = \{1, 2\}$ and for all $z \in Z$, $u_1(z) = -u_2(z)$, an extensive game is said to be **zero-sum**.*
- *For each player $i \in N$, a partition $\mathbf{I}_i$ of $H_i$ with the property that $A(h) = A(h')$ whenever $h$ and $h'$ are in the same member of the partition. $\mathbf{I}_i$ is the **information partition** of player $i$; a set $I_i \in \mathbf{I}_i$ is an **information set** of player $i$.*

In this paper, we exclusively focus on two-player zero-sum games with **perfect recall**, which is a restriction on the information partitions that excludes unrealistic situations where a player is forced to forget her own past information or decisions.

To play an extensive game each player specifies a strategy. A strategy determines how a player makes her decisions when confronted with a choice.

**Definition 2 (Strategy)** *A **strategy for player $i$**, $\sigma_i$, that assigns a probability distribution over $A(h)$ to each $h \in H_i$. This function is constrained so that $\sigma_i(h) = \sigma_i(h')$ whenever $h$ and $h'$ are in the same information set. A strategy is **pure** if no randomization is required. We denote $\Sigma_i$ as the set of all strategies for player $i$.*

**Definition 3 (Strategy Profile)** *A **strategy profile** in extensive game $\Gamma$ is a set of strategies, $\sigma = \{\sigma_1, \ldots, \sigma_n\}$, that contains one strategy for each player. We let $\sigma_{-i}$ denote the set strategies for all players except player $i$. We call the set of all strategy profiles $\Sigma$.*

When all players play according to a strategy profile, $\sigma$, we can define the expected utility of each player as $u_i(\sigma)$. Similarly, $u_i(\sigma_i, \sigma_{-i})$ is the expected utility of player $i$ when all other players play according to $\sigma_{-i}$ and player $i$ plays according to $\sigma_i$.

The traditional solution concept for extensive games is the Nash equilibrium concept.

**Definition 4 (Nash Equilibrium)** *A **Nash equilibrium** is a strategy profile $\sigma$ where*

$$\forall i \in N \ \forall \sigma_i' \in \Sigma_i \quad u_i(\sigma_i) \geq u_i(\sigma_i', \sigma_{-i}) \tag{1}$$

*An approximation of a Nash equilibrium or **$\varepsilon$-Nash equilibrium** is a strategy profile $\sigma$ where*

$$\forall i \in N \ \forall \sigma_i' \in \Sigma_i \quad u_i(\sigma_i) + \varepsilon \geq u_i(\sigma_i', \sigma_{-i}) \tag{2}$$

A Nash ($\varepsilon$-Nash) equilibrium is a strategy profile where no player can gain (more than $\varepsilon$) through unilateral deviation. A Nash equilibrium exists in all extensive games. For zero-sum extensive games with perfect recall we can efficiently compute an $\varepsilon$-Nash equilibrium using techniques such as linear programming [5], counterfactual regret minimization [9] and the excessive gap technique [2]. In a zero-sum game we say it is *optimal* to play any strategy belonging to an equilibrium because this guarantees the equilibrium player the highest expected utility in the worst case. Any deviation from equilibrium by either player can be exploited by a knowledgeable opponent. In this sense we can call computing an equilibrium in a zero-sum game *solving* the game.

Many games of interest are far too large to solve directly and abstraction is often employed to reduce the game to one of a more manageable size. The abstract game is solved and the resulting strategy is presumed to be strong in the original game. Abstraction can be achieved by merging information sets together, restricting the actions a player can take from a given history, or a combination of both.

**Definition 5 (Abstraction)** *[7] An **abstraction for player $i$** is a pair $\alpha_i = \langle \alpha_i^{\mathbf{I}}, \alpha_i^A \rangle$, where,*

- *$\alpha_i^{\mathbf{I}}$ is a partition of $H_i$, defining a set of abstract information sets coarser[1] than $\mathbf{I}_i$, and*
- *$\alpha_i^A$ is a function on histories where $\alpha_i^A(h) \subseteq A(h)$ and $\alpha_i^A(h) = \alpha_i^A(h')$ for all histories $h$ and $h'$ in the same abstract information set. We will call this the abstract action set.*

*The **null abstraction** for player $i$, is $\phi_i = \langle \mathbf{I}_i, A \rangle$. An **abstraction** $\alpha$ is a set of abstractions $\alpha_i$, one for each player. Finally, for any abstraction $\alpha$, the **abstract game**, $\Gamma^\alpha$, is the extensive game obtained from $\Gamma$ by replacing $\mathbf{I}_i$ with $\alpha_i^{\mathbf{I}}$ and $A(h)$ with $\alpha_i^A(h)$ when $P(h) = i$, for all $i$.*

Strategies for abstract games are defined in the same manner as for unabstracted games. However, the strategy must assign the same distribution to all histories in the same block of the abstraction's information partition, as well as assigning zero probability to actions not in the abstract action set.

## 3 Strategy Grafting

Though there is no guarantee that optimal strategies in abstract games are strong in the original game [7], these strategies have empirically been shown to perform well against both other computers [9] and humans [1]. Currently, strong strategies are solved for in one single equilibrium computation for a single abstract game. Advancement typically involves developing algorithmic improvements to equilibrium finding techniques in order to find solutions to yet larger abstract games.

It is simple to show that a strategy space must include at least as good, if not better, strategies than a smaller space that it refines [7]. At first glance, this would seem to imply that a larger abstraction would always be better, but upon closer inspection we see this depends on our method of selecting a strategy from the space. In poker, when using arbitrary equilibrium strategies that are evaluated in a tournament setting, this intuition empirically holds true.

One potentially important factor for the empirical evidence is the presence of dominated strategies in the support of the abstract equilibrium strategies.

**Definition 6 (Dominated Strategy)** *A **dominated strategy for player $i$** is a pure strategy, $\sigma_i$, such that there exists another strategy, $\sigma_i'$, where for all opponent strategies $\sigma_{-i}$,*

$$u_i(\sigma_i', \sigma_{-i}) \geq u_i(\sigma_i, \sigma_{-i}) \tag{3}$$

*and the inequality must hold strictly for at least one opponent strategy.*

This implies that a player can never benefit by playing a dominated strategy. When abstracting one can, in effect, merge a dominated strategy in with a non-dominated strategy. In the abstract game, this combined strategy might become part of an equilibrium and hence the abstract strategy would make occasional mistakes. That is, abstraction does not necessarily preserve strategy domination. As a result of their expressive power, finer abstractions may better preserve domination and thus can result in less play of dominated strategies.

Decomposition is a natural approach for using larger strategy spaces without incurring additional computational costs and indeed it has been employed toward this end. In extensive games with imperfect information, though, straightforward decomposition can be problematic. One way that equilibrium strategies guard against exploitation is information hiding, *i.e.*, the equilibrium plays in a fashion that hinders an opponent's ability to effectively reconstruct the player's private information. Independent solutions to a set of sub-games, though, may not "mesh", or hide information, effectively as a whole. For example, an observant opponent might be able to determine which sub-game is being played, which itself could be valuable information that could be exploited.

Armed with some intuition for why increasing the size of the strategy space may improve the quality of the solution and why decomposition can be problematic, we will now begin describing the strategy grafting algorithm and provide some theoretical results regarding the quality of grafted strategies. First, we will explain how a game of imperfect information is formally divided into sub-games.

**Definition 7 (Grafting Partition)** $G = \{G_0, G_1, \ldots, G_p\}$ *is a **grafting partition for player $i$** if*

1. *$G$ is a partition of $H_i$,*
2. *$\forall I \in \mathbf{I}_i \; \exists j \in \{0, \ldots, p\}$ such that $I \subseteq G_j$, and*
3. *$\forall j \in \{1, \ldots, p\}$ if $h$ is a prefix of $h' \in H_i$ and $h \in G_j$ then $h' \in G_j \cup G_0$.*

Using the elements of a grafting partition, we construct a set of sub-games. The solutions to these sub-games are called grafts, and we can combine them naturally, since they are disjoint sets, into one single grafted strategy.

**Definition 8 (Grafted Strategy)** *Given a strategy $\sigma_i \in \Sigma_i$ and a grafting partition $G$ for player $i$. For $j \in \{1, \ldots, p\}$, define $\Gamma^{\sigma_i,j}$ to be an extensive game derived from the original game $\Gamma$ where for all $h \in H_i \setminus G_j$, $P(h) = c$ and $f_c(a|h) = \sigma_i(h, a)$. That is, player $i$ only controls her actions for histories in $G_j$ and is forced to play according to $\sigma_i$ elsewhere. Let the **graft** of $G_j$, $\sigma^{*,j}$, be an $\epsilon$-Nash equilibrium of the game $\Gamma^{\sigma_i,j}$. Finally, define the **grafted strategy for player $i$** $\sigma_i^*$ as,*

$$\sigma_i^*(h, a) = \left\{ \begin{array}{ll} \sigma_i(h, a) & \text{if } h \in G_0 \\ \sigma_i^{*,j}(h, a) & \text{if } h \in G_j \end{array} \right.$$

*We will call $\sigma_i$ the **base strategy** and $G$ the grafting partition for the grafted strategy $\sigma_i^*$.*

There are a few key ideas to observe about grafted strategies that distinguish them from previous sub-game decomposition methods. First, we start out with a base strategy for the player. This base strategy can be constructed using current techniques for a tractably sized abstraction. It is important that we use the same base strategy for all grafts, as it is the only information that is shared between the grafts. Second, when we construct a graft, only the portion of the game that the graft plays is allowed to vary for our player of interest. The actions over the remainder of the game are played according to the base strategy. This allows us to refine the abstraction for that block of the grafting partition, so that it itself is as large as the largest tractably solvable game. Third, note that when we construct a graft, we continue to use an equilibrium finding technique, but we are not interested in the pair of strategies — we are only interested in the strategy for the player of interest. This means in games like poker, where we are interested in a strategy for both players, we must construct a grafted strategy separately for each player. Finally, when we construct a graft, our opponent must learn a strategy for the entire, potentially abstract, game. By letting our opponent's strategy vary completely, our graft will be a strategy that is less prone to exploitation, forcing each individual graft to mesh well with the base strategy and in turn with each other graft when combined.

Strategy grafting allows us to construct a strategy with more expressive power that what can be computed by solving a single game. We now show that strategy grafting uses this expressive power to its advantage, causing an (approximate) improvement over its base strategy. Note that we cannot guarantee a strict improvement as the base strategy may already be an optimal strategy.

**Theorem 1** *For strategies $\sigma_1, \sigma_2$ where $\sigma_2$ is an $\epsilon$-best response to $\sigma_1$, if $\sigma_1^*$ is the grafted strategy for player 1 where $\sigma_1$ is used as the base strategy and $G$ is the grafting partition then,*

$$u_1(\sigma_1^*, \sigma_2) - u_1(\sigma_1, \sigma_2) = \sum_{j=1}^{p} \left( u_1(\sigma_1^{*,j}, \sigma_2) - u_1(\sigma_1, \sigma_2) \right) \geq -3p\epsilon.$$

*In other words, the grafted strategy's improvement against $\sigma_2$ is equal to the sum of the gains of the individual grafts against $\sigma_2$ and this gain is no less than $-3p\epsilon$.*

PROOF. Define $Z_j$ as follows,

$$\forall j \in \{1, \ldots, p\} \quad Z_j = \{z \in Z \,|\, \exists h \in G_j \text{ with } h \text{ a prefix of } z\} \tag{4}$$

$$Z_0 = Z \setminus \bigcup_{j=1}^{p} Z_j \tag{5}$$

By condition (3) of Definition 7, $Z_{j=0,\ldots,p}$ are disjoint and therefore form a partition of $Z$.

$$\sum_{j=1}^{p} \left( u_1(\sigma_1^{*,j}, \sigma_2) - u_1(\sigma_1, \sigma_2) \right) \tag{6}$$

$$= \sum_{j=1}^{p} \left( \sum_{z \in Z} u_1(z) \Pr(z|\sigma_1^{*,j}, \sigma_2) - \sum_{z \in Z} u_1(z) \Pr(z|\sigma_1, \sigma_2) \right) \tag{7}$$

$$= \sum_{j=1}^{p} \sum_{k=0}^{p} \sum_{z \in Z_k} u_1(z) \left( \Pr(z|\sigma_1^{*,j}, \sigma_2) - \Pr(z|\sigma_1, \sigma_2) \right) \tag{8}$$

Notice that for all $z \in Z_{k \neq j}$, $\Pr(z|\sigma_1^{*,j}, \sigma_2) = \Pr(z|\sigma_1, \sigma_2)$, so only when $k = j$ is the summand non-zero.

$$= \sum_{j=1}^{p} \sum_{z \in Z_j} u_1(z) \left( \Pr(z|\sigma_1^{*,j}, \sigma_2) - \Pr(z|\sigma_1, \sigma_2) \right) \tag{9}$$

$$= \sum_{j=1}^{p} \sum_{z \in Z_j} u_1(z) \left( \Pr(z|\sigma_1^*, \sigma_2) - \Pr(z|\sigma_1, \sigma_2) \right) \tag{10}$$

$$= \sum_{z \in Z} u_1(z) \left( \Pr(z|\sigma_1^*, \sigma_2) - \Pr(z|\sigma_1, \sigma_2) \right) \tag{11}$$

$$= \left( \sum_{z \in Z} u_1(z) \Pr(z|\sigma_1^*, \sigma_2) - \sum_{z \in Z} u_1(z) \Pr(z|\sigma_1, \sigma_2) \right) \tag{12}$$

$$= u_1(\sigma_1^*, \sigma_2) - u_1(\sigma_1, \sigma_2) \tag{13}$$

Furthermore, since $\sigma_1^{*,j}$ and $\sigma_2^{*,j}$ are strategies of the $\epsilon$-Nash equilibrium $\sigma^{*,j}$,

$$u_1(\sigma_1^{*,j}, \sigma_2) + \epsilon \geq u_1(\sigma_1^{*,j}, \sigma_2^{*,j}) \geq u_1(\sigma_1, \sigma_2^{*,j}) - \epsilon \tag{14}$$

Moreover, because $\sigma_2$ is an $\epsilon$-best response to $\sigma_1$,

$$u_1(\sigma_1, \sigma_2^{*,j}) \geq u_1(\sigma_1, \sigma_2) - \epsilon \tag{15}$$

So, $\sum_{j=1}^{p} \left( u_1(\sigma_1^{*,j}, \sigma_2) - u_1(\sigma_1, \sigma_2) \right) \geq -3p\epsilon$. ∎

The main application of this theorem is in the following corollary, which follows immediately from the definition of an $\epsilon$-Nash equilibrium.

**Corollary 1** *Let $\alpha$ be an abstraction where $\alpha_2 = \phi_2$ and $\sigma$ be an $\epsilon$-Nash equilibrium strategy for the game $\Gamma^\alpha$, then any grafted strategy $\sigma_1^*$ in $\Gamma$ with $\sigma_1$ used as the base strategy will be at most $3p\epsilon$ worse than $\sigma_1$ against $\sigma_2$.*

Although these results suggest that a grafted strategy will (approximately) improve on its base strategy against an optimal opponent, there is one caveat: it assumes we know the opponent's abstraction or can solve a game with the opponent unabstracted. Without this knowledge or ability, this guarantee does not hold. However, all previous work that employs the use of abstract equilibrium strategies also implicitly makes this assumption. Though we know that refining an abstraction also has no guarantee on improving worst-case performance in the original game [7], the AAAI Computer Poker Competition [10] has shown that in practice larger abstractions and more expressive strategies consistently perform well in the original game, even though competition opponents are not using the same abstractions. We might expect a similar result even when the theorem's assumptions are not satisfied. In the next section we examine empirically both situations where we know our opponent's abstraction and situations where we do not.

## 4    Experimental Results

The AAAI Computer Poker Competitions use various types of large Texas Hold'em poker games. These games are quite large and the resulting abstract games can take weeks of computation to solve. We begin our experiments in a smaller poker game called Leduc Hold'em where we can examine several grafted strategies. This is followed by analysis of a grafted strategy for two-player limit Texas Hold'em that was submitted to the 2009 AAAI Poker Competition.

### 4.1    Leduc Hold'em

Leduc Hold'em is a two player poker game. The deck used in Leduc Hold'em contains six cards, two jacks, two queens and two kings, and is shuffled prior to playing a hand. At the beginning of a hand, each player pays a one chip ante to the pot and receives one private card. A round of betting then takes place starting with player one. After the round of betting, a single public card is revealed from the deck, which both players use to construct their hand. This card is called the *flop*. Another round of betting occurs after the flop, again starting with player one, and then a showdown takes place. At a showdown, if either player has paired their private card with the public card they win all the chips in the pot. In the event neither player pairs, the player with the higher card is declared the winner. The players split the money in the pot if they have the same private card.

Each betting round follows the same format. The first player to act has the option to *check* or *bet*. When betting the player adds chips into the pot and action moves to the other player. When a player faces a bet, they have the option to *fold*, *call* or *raise*. When folding, a player forfeits the hand and all the money in the pot is awarded to the opposing player. When calling, a player places enough chips into the pot to match the bet faced and the betting round is concluded. When raising, the player must put more chips into the pot than the current bet faced and action moves to the opposing player. If the first player checks initially, the second player may check to conclude the betting round or bet. In Leduc Hold'em there is a limit of one bet and one raise per round. The bets and raises are of a fixed size. This size is two chips in the first betting round and four chips in the second.

**Tournament Setup.**    Despite using a smaller poker game, we aim to create a tournament setting similar to the AAAI Poker Competition. To accomplish this we will create a variety of equilibrium-like players using abstractions of varying size. Each of these strategies will then be used as a base strategy to create two grafted strategies. All strategies are then played against each other in a round-robin tournament. A strategy is said to *beat* another strategy if its expected winnings against the other is positive. Unlike the AAAI Poker Competition, in our smaller game we can feasibly compute the expected value of one strategy against another and thus we are not required to sample.

The abstractions used are *J.Q.K*, *JQ.K*, and *J.QK*. Prior to the flop, the first abstraction can distinguish all three cards, the second abstraction cannot distinguish a jack from a queen and the third cannot distinguish a queen from a king. Postflop, all three abstractions are only aware of if they have paired their private card. These three abstractions were hand chosen as they are representative of how current abstraction techniques will group hands together. The first abstraction is the biggest, and hence we would expect it to do the best. The second and third abstractions are the same size.

We chose to train two types of grafted strategies: *preflop grafts* and *flop grafts*. Both types consist of three individual grafts for each player: one to play each card with complete information. That is,

| | (1) | (2) | (3) | (4) | (5) | (6) | (7) | (8) | (9) | Avg. |
|---|---|---|---|---|---|---|---|---|---|---|
| (1) J.Q.K preflop grafts | | 2.3 | 28.0 | 17.5 | 12.2 | 26.6 | 36.7 | 22.3 | 54.7 | 25.0 |
| (2) J.Q.K flop grafts | -2.3 | | 28.6 | 18.6 | 16.9 | 23.9 | 39.7 | 24.7 | 49.6 | 25.0 |
| (3) JQ.K flop grafts | -28.0 | -28.6 | | -47.2 | 67.0 | -0.9 | 28.5 | 79.9 | 89.2 | 20.0 |
| (4) JQ.K preflop grafts | -17.5 | -18.6 | 47.2 | | -11.2 | 9.0 | 67.3 | 3.7 | 62.8 | 17.9 |
| (5) J.QK preflop grafts | -12.2 | -16.9 | -67.0 | 11.2 | | 8.1 | -20.0 | 30.9 | 110.0 | 5.5 |
| (6) J.Q.K | -26.6 | -23.9 | 0.9 | -9.0 | -8.1 | | 13.6 | 7.5 | 32.5 | -1.6 |
| (7) JQ.K | -36.7 | -39.7 | -28.5 | -67.3 | 20.0 | -13.6 | | 42.2 | 70.6 | -6.6 |
| (8) J.QK flop grafts | -22.3 | -24.7 | -79.9 | -3.7 | -30.9 | -7.5 | -42.2 | | 83.3 | -16.0 |
| (9) J.QK | -54.7 | -49.6 | -89.2 | -62.8 | -110.0 | -32.5 | -70.6 | -83.3 | | -69.1 |

Table 1: Expected winnings of the row player against the column player in millibets per hand (mb/h)

| Strategy | Wins | Losses | Exploitability |
|---|---|---|---|
| J.Q.K preflop grafts | 8 | 0 | 298.3 |
| J.Q.K flop grafts | 7 | 1 | 321.1 |
| JQ.K preflop grafts | 5 | 3 | 465.9 |
| JQ.K flop grafts | 4 | 4 | 509.0 |
| J.QK preflop grafts | 4 | 4 | 507.3 |
| J.Q.K | 4 | 4 | 315.1 |
| JQ.K | 3 | 5 | 246.8 |
| J.QK flop grafts | 1 | 7 | 503.5 |
| J.QK | 0 | 8 | 371.1 |

Table 2: Each strategy's number of wins, losses, and exploitability in unabstracted Leduc Hold'em in millibets per hand (mb/h)

each graft does not abstract the sub-game for the observed card. These two types differ in that the preflop grafts play for the entire game whereas the flop grafts only play the game after the flop. For preflop grafts, this means $G_0$ is empty, *i.e.,* the final grafted strategy is always using the probabilities from some graft and never the base strategy. For flop grafts, the grafted strategy follows the base strategy in all preflop information sets. We use $\varepsilon$-Nash equilibria in the three abstract games as our base strategies. Each base strategy and graft is trained using counterfactual regret minimization for one billion iterations. The equilibria found are $\varepsilon$-Nash equilibria where no player can benefit more than $\varepsilon = 10^{-5}$ chips by deviating within the abstract game. We measure the expected winnings in *millibets per hand* or mb/h. A millibet is one thousandth of a small bet, or 0.002 chips.

**Results.** We can see in Table 1 that the grafted strategies perform well in a field of equilibrium-like strategies. The base strategy seems to be of great importance when training a grafted strategy. Though *JQ.K* and *J.QK* are the same size, the *JQ.K* strategy performs better in this tournament setting. Similarly, the grafted strategies appear to maintain the ordering of their base strategies either when considering the expected winnings in Table 1 or the number of wins in Table 2 (though *JQ.K flop grafts* switches places with *JQ.K preflop grafts* in the ordering). Although the choice of base strategy is important, the grafted strategies do well under both evaluation criteria and even the worst base strategy sees great relative improvement when used to train grafted strategies.

There are also a few other interesting trends in these results. First, our intuition that larger strategies perform better seems to hold in all cases except for *J.QK flop grafts*. Larger abstractions also perform better for the non-grafted strategies as *J.Q.K* is the biggest equilibrium strategy and it performs the best out of this group. Second, it appears that the preflop grafts are usually better than the flop grafts. This can be explained by the fact that the preflop grafts have more information about the original game. Finally, observe that the grafted strategies can have worse exploitability in the original game than their corresponding base strategy. Although this can make grafted strategies more vulnerable to exploitive strategies, they appear to perform well against a field of equilibrium-like opponents. In fact, in our experiment, grafted strategies appear to only improve upon the base strategy despite not always knowing the opponent's abstraction. This suggests that exploitability is not the only important measure of strategy quality. Contrast the grafted strategies with the strategy that always folds, which is exploitable at 500 mb/h. Although always folding is less exploitable than some of the grafted strategies, it cannot win against any opponent and would place last in this tournament.

|  | Relative Size | (1) | (2) | (3) | (4) | (5) | (6) | Avg. |
|---|---|---|---|---|---|---|---|---|
| **(1) 20x8 Grafted** | 1.0 |  | 2.1 | 14.5 | 18.1 | 13.7 | 18.7 | 13.4 |
| **(2) 20x32** | 2.53 | -2.1 |  | 4.9 | 9.4 | 11.8 | 15.5 | 7.9 |
| **(3) 20x8 (Base)** | 1.0 | -14.5 | -4.9 |  | 6.2 | 7.2 | 10.7 | 0.9 |
| **(4) 20x7** | 0.43 | -18.1 | -9.4 | -6.2 |  | 1.7 | 5.0 | -5.4 |
| **(5) 14** | 0.82 | -13.7 | -11.8 | -7.2 | -1.7 |  | 5.3 | -5.8 |
| **(6) 12** | 0.45 | -18.7 | -15.5 | -10.7 | -5.0 | -5.3 |  | -11.0 |

Table 3: Sampled expected winnings in Texas Hold'em of the row player against the column player in millibets per hand (mb/h). 95% confidence intervals are between $0.8$ and $1.6$. Relative size is the ratio of the size of the abstract game(s) solved for the row strategy and the base strategy.

## 4.2 Texas Hold'em

Two-player limit Texas Hold'em bears many similarities to Leduc Hold'em but is much larger in scale with respect to the parameters: cards in the deck, private cards, public cards, betting rounds and bets per round. Due to the computational cost[2] needed to solve a strong equilibrium, our experiments consist of a single grafted strategy. Table 3 shows the results of running this large grafted strategy against equilibrium-like strategies using a variety of abstractions.

The *20x32* strategy is the largest single imperfect recall abstract game solved to date. It is approximately $2.53$ times larger than the base strategy used with grafting, *20x8*. The *20x7* (imperfect recall) and *12* (perfect recall) strategies were the entrants put forward by the Computer Poker Research Group for the 2008 and 2007 AAAI Computer Poker Competitions, respectively. The *14* strategy was considered for the 2008 competition, but it was ultimately superseded by the smaller *20x7*. For a detailed description of these abstractions and the rules of Texas Hold'em see *A Practical Use of Imperfect Recall* [8].

As evident in the results, the grafted strategy beats all of the players with statistical significance, even the largest single strategy. In addition to these results against other Computer Poker Research Group strategies, the grafted strategy also performed well at the 2009 AAAI Computer Poker Competition. There, against a field of thirteen strong strategies, it placed second and fourth (narrowly behind the third place entrant) in the limit run-off and limit bankroll competitions, respectively.

These results demonstrate that strategy grafting is competitive and allows one to augment their existing strategies. Any improvement to the quality of a base strategy should in turn improve the quality of the grafted strategy in similar tournament settings. This means that strategy grafting can be used transparently on top of more sophisticated strategy-computing methods.

## 5 Conclusion

We have introduced a new method, called strategy grafting, for independently solving and combining sub-games in large extensive games. This method allows us to create larger strategies than previously possible by solving many sub-games. These new strategies seem to maintain the features of good equilibrium-like strategies. By creating larger strategies we hope to play fewer dominated strategies and, in turn, make fewer mistakes. Against a static equilibrium-like opponent, making fewer mistakes should lead to an improvement in the quality of play. Our empirical results confirm this intuition and demonstrate that this new method can improve the performance of the state-of-the-art in both a simulated competition and the actual AAAI Computer Poker Competition. It is likely that much of the strength of these new strategies will be bounded by the quality of the base strategy used. In this regard, we are still limited by the capabilities of current methods.

## Acknowledgments

The authors would like to thank the members of the Computer Poker Research Group at the University of Alberta for helpful conversations pertaining to this research. This research was supported by NSERC, iCORE, and Alberta Ingenuity.

## Footnotes

[1]Partition $A$ is coarser than partition $B$, if and only if every set in $B$ is a subset of some set in $A$, or equivalently $x$ and $y$ are in the same set in $A$ if $x$ and $y$ are in the same set in $B$.

[2]This particular grafted strategy was computed on a large cluster using 640 processors over almost 6 days.

# References

[1] Darse Billings, Neil Burch, Aaron Davidson, Robert Holte, Jonathan Schaeffer, Terance Schauenberg, and Duane Szafron. Approximating Game-Theoretic Optimal Strategies for Full-scale Poker. In *International Joint Conference on Artificial Intelligence*, pages 661–668, 2003.

[2] Andrew Gilpin, Samid Hoda, Javier Peña, and Tuomas Sandholm. Gradient-based Algorithms for Finding Nash Equilibria in Extensive Form Games. In *Proceedings of the Eighteenth International Conference on Game Theory*, 2007.

[3] Andrew Gilpin and Tuomas Sandholm. A Competitive Texas Hold'em Poker Player via Automated Abstraction and Real-time Equilibrium Computation. In *Proceedings of the Twenty-First Conference on Artificial Intelligence*, 2006.

[4] Andrew Gilpin and Tuomas Sandholm. Expectation-Based Versus Potential-Aware Automated Abstraction in Imperfect Information Games: An Experimental Comparison Using Poker. In *Proceedings of the Twenty-Third Conference on Artificial Intelligence*, 2008.

[5] Daphne Koller and Avi Pfeffer. Representations and Solutions for Game-Theoretic Problems. *Artificial Intelligence*, 94:167–215, 1997.

[6] Martin Osborne and Ariel Rubinstein. *A Course in Game Theory*. The MIT Press, Cambridge, Massachusetts, 1994.

[7] Kevin Waugh, David Schnizlein, Michael Bowling, and Duane Szafron. Abstraction Pathologies in Extensive Games. In *Proceedings of the Eighth International Joint Conference on Autonomous Agents and Multi-Agent Systems*, pages 781–788, 2009.

[8] Kevin Waugh, Martin Zinkevich, Michael Johanson, Morgan Kan, David Schnizlein, and Michael Bowling. A Practical Use of Imperfect Recall. In *Proceedings of the Eighth Symposium on Abstraction, Reformulation and Approximation*, 2009.

[9] Martin Zinkevich, Michael Johanson, Michael Bowling, and Carmelo Piccione. Regret Minimization in Games with Incomplete Information. In *Advances in Neural Information Processing Systems Twenty*, pages 1729–1736, 2008. A longer version is available as a University of Alberta Technical Report, TR07-14.

[10] Martin Zinkevich and Michael Littman. The AAAI Computer Poker Competition. *Journal of the International Computer Games Association*, 29, 2006. News item.

